# Signal Processing by Multiplexing and Demultiplexing in Neurons

David C. Tam
Division of Neuroscience
Baylor College of Medicine
Houston, TX 77030
dtam@next-cns.neusc.bcm.tmc.edu

## Abstract

Signal processing capabilities of biological neurons are investigated. Temporally coded signals in neurons can be multiplexed to increase the transmission capacity. Multiplexing of signal is suggested in bi-threshold neurons with "high-threshold" and "low-threshold" for switching firing modes. To extract the signal embedded in the interspike-intervals of firing, the encoded signal are demultiplexed and multiplexed by a network of neurons with delayed-line circuitry for signal processing. The temporally coded input signal is transformed spatially by mapping the firing intervals topographically to the output of the network, thus decoding the specific firing interspike-intervals. The network also provides a band-pass filtering capability where the variability of the timing of the original signal can be decoded.

## 1 INTRODUCTION

Signals of biological neurons are encoded in the firing patterns of spike trains or the time series of action potentials generated by neurons. The signal content of the codes encoded by a presynaptic neuron will be decoded by some other neurons postsynpatically. Neurons are often thought to be encoding a single type of

codes. But there is evidence suggesting that neurons may encode more than one type of signals. One of the mechanisms for embedding multiple types of signals processed by a neuron is multiplexing. When the signals are multiplexed, they also need to be demultiplexed to extract the useful information transmitted by the neurons. Theoretical and experimental evidence of such multiplexing and demultiplexing scheme for signal processing by neurons will be given below.

## 2 MULIPLEXING IN NEURONS

Most neurons fire action potentials when the membrane potential is depolarized to a threshold above the resting potential. For some neurons, there are more than a single threshold that can trigger the generation of action potentials. The thresholds occur not only at depolarized membrane potential (above the resting potential) but also at hyperpolarized potential (below the resting potential). This bi-threshold phenomena had been reported in a number of biological neurons including the giant squid axon (Hodgkin & Huxley, 1952), thalamic (Jahnsen & Llinás, 1984), inferior olivary (Yarom & Llinás, 1987), and hippocampal neurons (Stasheff & Wilson, 1990). The phenomena of triggering the firing of action potentials at a membrane potential below the resting potential level following prolonged hyperpolarization have been observed under different conditions in different neurons such as during the anodal break after voltage-clamped at a hyperpolarized potential (Hodgkin & Huxley, 1952), and are called "low-threshold spikes" (Yarom & Llinás, 1987) and "baseline spikes" (Stasheff & Wilson, 1990), which are spikes elicited naturally during the after-hyperpolarization (a.h.p.) period. The generation of low-threshold spikes is a voltage- and time-dependent process occurring during a prolonged hyperpolarization for de-inactivation of ionic conductances.

Given this bi-threshold for firing of action potentials, a neuron can function in two modes of operations: one at depolarization potentials and the other at hyperpolarization potentials. Thus, when the neuron is depolarized from the resting potential, the neuron will process signal based on the "high-threshold", and when the neurons is hyperpolarized for a prolonged duration, the neuron will process signal based on the "low-threshold". Formally, it is described as follows:

$$y(t) = \begin{cases} 1, & if\ V(t) \geq \theta_{hi} \qquad\qquad or \\ & if\ V(t-i\Delta t) < \theta_{lo}\ and\ V(t) \geq \theta_{lo},\ for\ 1 < i < j \\ 0, & otherwise \end{cases} \qquad (1)$$

where $y(t)$ denotes the occurrence of the firing of an action potential at time $t$, $x(t)$ denotes the membrane potential of the neuron at time $t$, $\theta_{hi}$ denotes the "high-threshold" and $\theta_{lo}$ denotes the "low-threshold", and $j\Delta t$ represents the duration of hyperpolarization, such that the neuron will fire when

depolarized at the hyperpolarization potential. This bi-threshold firing phenomenon was suggested to be involved in the two different rhythms generated by a neuron as a periodic bi-stable oscillator (Rose & Hindmarsh, 1985; Goldbeter & Moran, 1988), which can switch between two different firing frequencies, thus multiplexing the signal depending on the mode of operation or polarization level (Tam, 1990c).

## 3 DEMULTIPLEXING IN NEURONS

The multiplexed signal encoded in a neuron can be demultiplexed in a number of ways. One of the systematic way of extracting the firing frequency of the encoded signal can be described by a network of neurons. Given the temporally modulated input spike train spike, the firing intervals of the encoded signal can be extracted by a network of neurons such that the firing of these output neurons will decode the interspike-intervals of the input signal. In this network, the temporal codes of the input spike train will be converted into a spatially-distributed topographical code where each output neuron represents a particular firing interval with a specific band-width. Thus, the original signal is demultiplexed by mapping the input firing intervals into the firing of specific neurons based on the spatial location of the neuron in the output layer.

The circuitry of this network of neurons utilizes delay-lines for signal processing (Reiss, 1964; Tam, 1990a, b). Examples of delay-line architecture used for signal processing can be found in the cerebellar cortex (Eccles *et al.*, 1967), inferior colliculus (Yin, *et al.*, 1987, 1986, 1985; Chan *et al.*, 1987) and cochlear nucleus (Carr & Konishi, 1990).

The time-delayed network can be described as follows. Let $x(t)$ be a time-series of spikes (or delta-functions, $\delta(t)$) with a total of $n+1$ spikes:

$$x(t) = \sum_{j=0}^{n} \delta(t - \tau_j) \tag{2}$$

Let the input to the network be a spike train $x(t)$ given by (2). There are $k$ neurons in the first input layer of the network. The input is split into multiple branches, each of which is connected to all $k$ neurons in the first layer. In addition to the direct connection between the input and the first layer neurons, each input branch to the first layer neuron is also split into multiple branches with successive incremental time-delays. Specially, the $k$-th neuron in the first layer has $k+1$ input lines, each input is successively delayed by a time delay $\Delta t$ relative to the previous one. That is, the $i$-th input to this $k$-th neuron in the first layer at time $t$ is given by $x(t-i\Delta t)$. Thus, the sum of the input to this $k$-th neuron is given by:

$$X_k(t) = \sum_{i=0}^{k} x(t-i\Delta t) \tag{3}$$

## 3.1 BAND-PASS FILTERING

Band-pass filtering can be accomplished by the processing at the first layer of neurons. If the threshold for the generation of an output spike for the $k$-th neuron is set at one, then this neuron will fire only when the interspike-interval, $I_j$, of the input spike train is within the time-delay window, $k\Delta t$. That is, the output of this $k$-th neuron is given by:

$$y_k(t) = \begin{cases} 1, & if \ X_k > 1 \\ 0, & otherwise \end{cases} \tag{4}$$

The *interspike-interval*, $I_j$, is defined as the time interval between any two adjacent spikes:

$$I_j = \tau_j - \tau_{j-1} \quad for \ 0 < j \le n \tag{5}$$

Therefore, the $k$-th neuron can be considered as encoding a band-pass filtered input interspike-interval, $0 < I_j \le k\Delta t$. Thus, the $k$-th neuron in the first layer essentially capture the input interspike-interval firing of less than $k\Delta t$, the band-passed interspike-interval. To ensure that the neuron will fire a spike of $\Delta t$ in duration, we introduce a refractory period of $(k-1)\Delta t$ after the firing of a spike for the $k$-th neuron to suppress continual activation of the neuron due to the phase differences of the incoming delayed signal.

## 3.2 HIGHER-ORDER INTERSPIKE-INTERVAL PROCESSING

Higher-order interspike-intervals can be eliminated by the second layer neurons. The *order* of the interspike-interval is defined by the number of intervening spikes between any two spikes in the spike train. That is, the first-order interspike-interval contains no intervening spike between the two adjacent spikes under consideration. Second-order interspike-interval is the time interval between two consecutive first-order interspike-intervals, i.e., the interval containing one intervening spike.

If the second layer neurons receive excitatory input from the corresponding neuron with a threshold ($\theta > 1$) and inhibitory input from the corresponding neuron with a threshold of ($\theta > 2$), then the higher-order intervals are eliminated, with the output of the second layer (double-primed) neuron given by:

$$y_k''(t) = y_k(t) - y_k'(t) = \begin{cases} 1, & if \ 2 \ge X_k(t) > 1 \\ 0, & otherwise \end{cases} \tag{6}$$

where

$$y'_k(t) = \begin{cases} 1, & if\ X_k > 2 \\ 0, & otherwise \end{cases} \qquad (7)$$

This requires that an addition input layer of neurons be added to the network, which we call the *first-parallel layer*, whose input/output relationship is given by (7). In other words, there are $k$ first layer neurons and $k$ first-parallel layer neurons serving as the input layers of the network. The $k$-th neuron in the first layer and the $k$-th neuron in the first-parallel layer are similar in their inputs, but the thresholds for producing an output spike are different. The difference between the outputs of the first set of neurons (first layer) in the first layer and the primed set of neurons (first-parallel layer) is computed by the *second* layer by making excitatory connection from the first layer neuron and inhibitory connection from the first-parallel layer neuron for each corresponding $k$-th neuron respectively as described by (6). This will ensure accurate estimation of only first-order interspike-interval, $0 < I_j \leq k\Delta t$, within the time-delay window $k\Delta t$.

## 3.3 BAND-WIDTH PROCESSING

The third layer neurons will filter the input signal by distributing the frequency (or interval) of firing of neurons within a specific band-width. Since the $k$-th neuron in the second layer detects the band-passed first-order interspike-intervals $(0 < I_j \leq k\Delta t)$ and the $h$-th neuron detects another band-passed interspike-intervals $(0 < I_j \leq h\Delta t)$, then the difference between these two neurons will detect first-order interspike-intervals with a band-width of $(k-h)\Delta t$. In order words, it will detect the first-order interspike-interval between $k\Delta t$ and $h\Delta t$, i.e., $h\Delta t < I_j \leq k\Delta t$.

This requires that the *third* layer neurons derive their inputs from two sources: one excitatory and the other inhibitory from the second layer. The output of the $k$-th neuron in the third layer, $y'''_k(t)$, is obtained from the difference between the outputs of $k$-th and $h$-th neurons in the second layer:

$$y'''_{kh}(t) = y''_k(t) - y''_h(t) = \begin{cases} 1, & if\ 2 \geq \displaystyle\sum_{i=h}^{k} x(t-i\Delta t) > 1 \\ \\ 0, & otherwise \end{cases} \qquad (7)$$

A two-dimensional topographical map of the band-passed interspike-intervals of the input spike train can be represented by arranging the third-layer neurons in a two-dimensional array, with one axis (the horizontal axis) representing the $k$ index (the band-passed interspike-interval) of equation (7) and the other axis (the vertical axis) representing the $(k-h)$ index (the band-width

interspike-interval). Thus the firing of the third layer neurons represents the band-passed filtered version of the original input spike train, extracting the firing interspike-interval of the input signal. The "coordinate" of the neuron in the third layer represents the band-passed interspike-interval $(0 < I_j \leq k\Delta t)$ and the band-width interspike-interval $(h\Delta t < I_j \leq k\Delta t)$ of the original input spike train signal. The band-width can be used to detect the variations (or jittering) in the timing for firing of spikes in the input spike train, since the timing of firing of spikes in biological neurons can be very variable. Thus, the network can be used to detect the variability of timing in firing of spikes by the firing location of the third layer neuron.

## 3.4 EXTRACTION OF EMBEDDED SIGNAL BY BI-THRESHOLD FIRING

If the neurons in the second and third layers are bi-threshold neurons where one threshold is at the "depolarization" level (i.e., a positive value) and the other threshold is at the "hyperpolarization" level (i.e., a negative value), then addition information may be extracted based on the level of firing threshold. Since the neuron in the second and third layers receive inhibitory inputs from the preceding layer, there are instances where the neuron be "hyperpolarized" or the sum of the inputs to the neuron is negative. Such condition occurs when the order of the interspike-interval is higher than one. In other words, the higher-order interspike-interval signal is embedded in the "hyperpolarization", which is normally suppressed from generating a spike when there is only one threshold for firing at the "depolarized" level $(\theta_{hi})$. But for bi-threshold neurons where there is another threshold at the hyperpolarized level $(\theta_{lo})$, such embedded signal encoded as hyperpolarization can be extracted by sending an external depolarizing signal to this neuron causing the neuron to fire at the low threshold. Thus the hyperpolarization signal can be "read-out" by an external input to the bi-threshold neuron. In summary, a time-delay network can be used to process temporally modulated pulsed-coded spike train signal and extract the firing interspike-intervals by mapping the band-passed intervals topographically on a two-dimensional output array from which the order of the interspike-interval can be extracted using different thresholds of firing.

## Acknowledgements

This work is supported by ONR contract N00014-90-J-1353.

## References

Carr, C. E. & Konishi, M. (1990)  A circuit for detection of interaural time differences in the brain stem of the barn owl. *J. Neurosci.* 10: 3227-3246.

Chan, J. C., Yin, T. C. & Musicant, A. D. (1987)  Effects of interaural time delays of noise stimuli on low-frequency cells in the cat's inferior colliculus. II. Responses to band-pass filtered noises. *J. Neurophysiol.* 58: 543-561.

Goldbeter, A. & Moran, F. (1988)  Dynamics of a biochemical system with multiple oscillatory domains as a clue for multiple modes of neuronal oscillations. *Eur. Biophys. J.* 15:277-287.

Hodgkin, A. L. & Huxley, A. F. (1952) A quantitative description of membrane current and its application to conduction and excitation in nerve. *J. Physiol. (London)* 117: 500-544.

Eccles, J.C., Ito, M. and Szentágothai, J. (1967) *The Cerebellum as a Neuronal Machine*, Springer-Verlag, New York, Heidelberg.

Jahnsen, H. & Llinás, R. (1984)  Electrophysiological properties of guinea-pig thalamic neurones: An *in vitro* study. *J. Physiol. (London)* 349:205-226.

Reiss, R.F. (1964)  A theory of resonant networks.  In (Ed. R.F. Reiss) *Neural Theory and Modeling: Proceedings of the 1962 Ojai Symposium.* Stanford University Press, Stanford, CA.

Rose, R. M. & Hindmarsh, J. L. (1985)  A model of a thalamic neuron. *Proc. R. Soc. Lond.* 225:161-193.

Stasheff, S. F. & Wilson, W. A. (1990)  Increased ectopic action potential generation accompanies epileptogenesis in vitro. *Neurosci. Lett.* 111: 144-150.

Tam, D. C. (1990a)  Temporal-spatial coding transformation: Conversion of frequency-code to place-code via a time-delayed neural network. *Proceedings of the International Joint Conference on Neural Networks* (H. Caudill, eds.), *Jan., 1990.* Vol. 1, pp. I-130–133.

Tam, D. C. (1990b) Decoding of firing intervals in a temporal-coded spike train using a topographically mapped neural network. *Proc. of International Joint Conference on Neural Networks.* Vol. 3, pp. III-627–632.

Tam, D. C. (1990c) Functional significance of bi-threshold firing of neurons. *Society for Neuroscience Abstract.* Vol. 16, p. 1091.

Yarom, Y. & Llinás, R. (1987)  Long-term modifiability of anomalous and delayed rectification in guinea pig inferior olivary neurons. *J. Neurosci.* 7:1166-1177.

Yin, T. C., Chan, J. C. & Carney, L. H. (1987)  Effects of interaural time delays of noise stimuli on low-frequency cells in the cat's inferior colliculus. III.  Evidence for cross-correlation. *J. Neurophysiol.* 58: 562-583.

Yin, T. C., Chan, J. C. & Irvine, D. R. (1986)  Effects of interaural time delays of noise stimuli on low-frequency cells in the cat's inferior colliculus. I. Responses to wideband noise. *J. Neurophysiol.* 55: 280-300.

Yin, T. C., Hirsch, J. A. & Chan, J. C. (1985)  Responses of neurons in the cat's superior colliculus to acoustic stimuli.  II.  A model of interaural intensity sensitivity. *J. Neurophysiol.* 53: 746-758.